# ANALYSIS AND COMPARISON OF DIFFERENT LEARNING ALGORITHMS FOR PATTERN ASSOCIATION PROBLEMS

J. Bernasconi
Brown Boveri Research Center
CH-5405 Baden, Switzerland

## ABSTRACT

We investigate the behavior of different learning algorithms for networks of neuron-like units. As test cases we use simple pattern association problems, such as the XOR-problem and symmetry detection problems. The algorithms considered are either versions of the Boltzmann machine learning rule or based on the backpropagation of errors. We also propose and analyze a generalized delta rule for linear threshold units. We find that the performance of a given learning algorithm depends strongly on the type of units used. In particular, we observe that networks with ±1 units quite generally exhibit a significantly better learning behavior than the corresponding 0,1 versions. We also demonstrate that an adaption of the weight-structure to the symmetries of the problem can lead to a drastic increase in learning speed.

## INTRODUCTION

In the past few years, a number of learning procedures for neural network models with hidden units have been proposed[1,2]. They can all be considered as strategies to minimize a suitably chosen error measure. Most of these strategies represent local optimization procedures (e.g. gradient descent) and therefore suffer from all the problems with local minima or cycles. The corresponding learning rates, moreover, are usually very slow.

The performance of a given learning scheme may depend critically on a number of parameters and implementation details. General analytical results concerning these dependences, however, are practically non-existent. As a first step, we have therefore attempted to study empirically the influence of some factors that could have a significant effect on the learning behavior of neural network systems.

Our preliminary investigations are restricted to very small networks and to a few simple examples. Nevertheless, we have made some interesting observations which appear to be rather general and which can thus be expected to remain valid also for much larger and more complex systems.

## NEURAL NETWORK MODELS FOR PATTERN ASSOCIATION

An artificial neural network consists of a set of interconnected units (formal neurons). The state of the i-th unit is described by a variable $S_i$ which can be discrete (e.g. $S_i = 0,1$ or $S_i = \pm 1$) or continuous (e.g. $0 \leq S_i \leq 1$ or $-1 \leq S_i \leq +1$), and each connection $j \rightarrow i$ carries a weight $W_{ij}$ which can be positive, zero, or negative.

The dynamics of the network is determined by a local update rule,

$$S_i(t+1) = f(\sum_j W_{ij} S_j(t)) \quad , \tag{1}$$

where f is a nonlinear activation function, specifically a threshold function in the case of discrete units and a sigmoid-type function, e.g.

$$f(x) = 1/(1+e^{-x}) \tag{2}$$

or

$$f(x) = (1-e^{-x})/(1+e^{-x}) \quad , \tag{3}$$

respectively, in the case of continuous units. The individual units can be given different thresholds by introducing an extra unit which always has a value of 1.

If the network is supposed to perform a pattern association task, it is convenient to divide its units into input units, output units, and hidden units. Learning then consists in adjusting the weights in such a way that, for a given input pattern, the network relaxes (under the prescribed dynamics) to a state in which the output units represent the desired output pattern.

Neural networks learn from examples (input/output pairs) which are presented many times, and a typical learning procedure can be viewed as a strategy to minimize a suitably defined error function F. In most cases, this strategy is a (stochastic) gradient descent method: To a clamped input pattern, randomly chosen from the learning examples, the network produces an output pattern $\{O_i\}$. This is compared with the desired output, say $\{T_i\}$, and the error $F(\{O_i\}, \{T_i\})$ is calculated. Subsequently, each weight is changed by an amount proportional to the respective gradient of F,

$$\Delta W_{ij} = -\eta \frac{\partial F}{\partial W_{ij}} \quad , \tag{4}$$

and the procedure is repeated for a new learning example until F is minimized to a satisfactory level.

In our investigations, we shall consider two different types of learning schemes. The first is a deterministic version of the Boltzmann machine learning rule[1] and has been proposed by Yann Le Cun[2]. It applies to networks with symmetric weights, $W_{ij} = W_{ji}$, so that an energy

$$E(\underline{S}) = - \sum_{(i,j)} W_{ij} S_i S_j \tag{5}$$

can be associated with each state $\underline{S} = \{S_i\}$. If $\underline{X}$ refers to the network state when only the input units are clamped and $\underline{Y}$ to the state when both the input and output units are clamped, the error function

is defined as

$$F = E(\underline{Y}) - E(\underline{X}) \quad , \tag{6}$$

and the gradients are simply given by

$$-\frac{\partial F}{\partial W_{ij}} = Y_i Y_j - X_i X_j \quad . \tag{7}$$

The second scheme, called backpropagation or generalized delta rule[1,3], probably represents the most widely used learning algorithm. In its original form, it applies to networks with feedforward connections only, and it uses gradient descent to minimize the mean squared error of the output signal,

$$F = \frac{1}{2} \sum_i (T_i - O_i)^2 \quad . \tag{8}$$

For a weight $W_{ij}$ from an (input or hidden) unit j to an output unit i, we simply have

$$-\frac{\partial F}{\partial W_{ij}} = (T_i - O_i)f'(\sum_k W_{ik} S_k)S_j \quad , \tag{9}$$

where f' is the derivative of the nonlinear activation function introduced in Eq. (1), and for weights which do not connect to an output unit, the gradients can successively be determined by applying the chain rule of differentiation.

In the case of discrete units, f is a threshold function, so that the backpropagation algorithm described above cannot be applied. We remark, however, that the perceptron learning rule[4],

$$\Delta W_{ij} = \varepsilon(T_i - O_i)S_j \quad , \tag{10}$$

is nothing else than Eq. (9) with f' replaced by a constant $\varepsilon$. Therefore, we propose that a generalized delta rule for linear threshold units can be obtained if f' is replaced by a constant $\varepsilon$ in all the backpropagation expressions for $\partial F/\partial W_{ij}$. This generalization of the perceptron rule is, of course, not unique. In layered networks, e.g., the value of the constant which replaces f' need not be the same for the different layers.

## ANALYSIS OF LEARNING ALGORITHMS

The proposed learning algorithms suffer from all the problems of gradient descent on a complicated landscape. If we use small weight changes, learning becomes prohibitively slow, while large weight changes inevitably lead to oscillations which prevent the algorithm from converging to a good solution. The error surface, moreover, may contain many local minima, so that gradient descent is not guaranteed to find a global minimum.

There are several ways to improve a stochastic gradient descent procedure. The weight changes may, e.g., be accumulated over a number of learning examples before the weights are actually changed. Another often used method consists in smoothing the weight changes by overrelaxation,

$$\Delta W_{ij}(k+1) = -\eta \, \frac{\partial F}{\partial W_{ij}} + \alpha \, \Delta W_{ij}(k) \quad , \tag{11}$$

where $\Delta W_{ij}(k)$ refers to the weight change after the presentation of the k-th learning example (or group of learning examples, respectively). The use of a weight decay term,

$$\Delta W_{ij} = -\eta \, \frac{\partial F}{\partial W_{ij}} - \beta W_{ij} \quad , \tag{12}$$

prevents the algorithm from generating very large weights which may create such high barriers that a solution cannot be found in reasonable time.

Such smoothing methods suppress the occurrence of oscillations, at least to a certain extent, and thus allow us to use higher learning rates. They cannot prevent, however, that the algorithm may become trapped in bad local minimum. An obvious way to deal with the problem of local minima is to restart the algorithm with different initial weights or, equivalently, to randomize the weights with a certain probability p during the learning procedure. More sophisticated approaches involve, e.g., the use of hill-climbing methods.

The properties of the error-surface over the weight space not only depend on the choice of the error function F, but also on the network architecture, on the type of units used, and on possible restrictions concerning the values which the weights are allowed to assume.

The performance of a learning algorithm thus depends on many factors and parameters. These dependences are conveniently analyzed in terms of the behavior of an appropriately defined learning curve. For our small examples, where the learning set always consists of all input/output cases, we have chosen to represent the performance of a learning procedure by the fraction of networks that are "perfect" after the presentation of N input patterns. (Perfect networks are networks which for every input pattern produce the correct output). Such learning curves give us much more detailed information about the behavior of the system than, e.g., averaged quantities like the mean learning time.

## RESULTS

In the following, we shall present and discuss some representative results of our empirical study. All learning curves refer to a set of 100 networks that have been exposed to the same learning procedure, where we have varied the initial weights, or the sequence

of learning examples, or both. With one exception (Figure 4), the sequences of learning examples are always random.

A prototype pattern association problem is the exclusive-or (XOR) problem. Corresponding networks have two input units and one output unit. Let us first consider an XOR-network with only one hidden unit, but in which the input units also have direct connections to the output unit. The weights are symmetric, and we use the deterministic version of the Boltzmann learning rule (see Eqs. (5) to (7)). Figure 1 shows results for the case of tabula rasa initial conditions, i.e. the initial weights are all set equal to zero. If the weights are changed after every learning example, about 2/3 of the networks learn the problem with less than 25 presentations per pattern (which corresponds to a total number of $4 \times 25 = 100$ presentations). The remaining networks (about 1/3), however, never learn to solve the XOR-problem, no matter how many input/output cases are presented. This can be understood by analyzing the corresponding evolution-tree in weight-space which contains an attractor consisting of 14 "non-perfect" weight-configurations. The probability to become trapped by this attractor is exactly 1/3. If the weight changes are accumulated over 4 learning examples, no such attractor

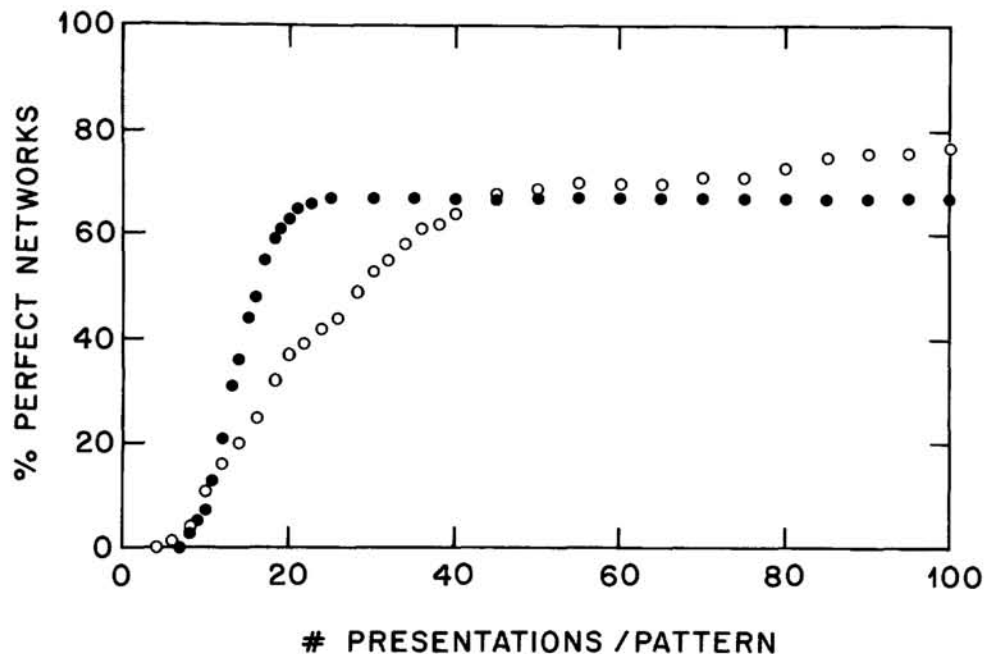

Fig. 1. Learning curves for an XOR-network with one hidden unit (deterministic Boltzmann learning, discrete ±1 units, initial weights zero). Full circles: weights changed after every learning example; open circles: weight changes accumulated over 4 learning examples.

seems to exist (see Fig. 1), but for some networks learning at least takes an extremely long time. The same saturation effect is observed with random initial weights (uniformly distributed between -1 and +1), see Fig. 2.

Figure 2 also exhibits the difference in learning behavior between networks with ±1 units and such with 0,1 units. In both cases, weight randomization leads to a considerably improved learning behavior. A weight decay term, by the way, has the same effect. The most striking observation, however, is that ±1 networks learn much faster than 0,1 networks (the respective average learning times differ by about a factor of 5). In this connection, we should mention that $\eta = 0.1$ is about optimal for 0,1 units and that for ±1 networks the learning behavior is practically independent of the value of $\eta$. It therefore seems that ±1 units lead to a much more well-behaved error-surface than 0,1 units. One can argue, of course, that a discrete 0,1 model can always be translated into a ±1 model, but this would lead to an energy function which has a considerably more complicated weight dependence than Eq. (5).

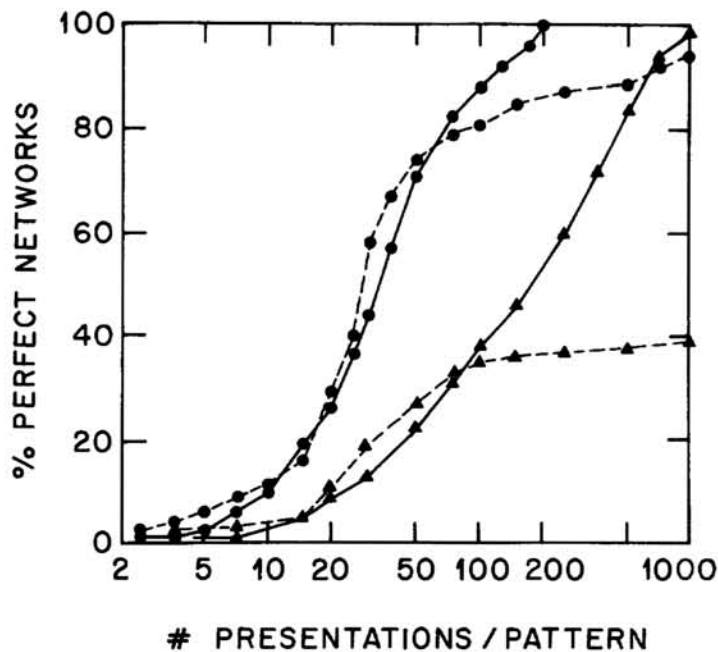

Fig. 2. Learning curves for an XOR-network with one hidden unit (deterministic Boltzmann learning, initial weights random, weight changes accumulated over 5 learning examples). Circles: discrete ±1 units, $\eta = 1$; triangles: discrete 0,1 units, $\eta = 0.1$; broken curves: without weight randomization; solid curves: with weight randomization ($p = 0.025$).

Figures 3 and 4 refer to a feedforward XOR-network with 3 hidden units, and to backpropagation or generalized delta rule learning. In all cases we have included an overrelaxation (or momentum) term with $\alpha = 0.9$ (see Eq. (11)). For the networks with continuous units we have used the activation functions given by Eqs. (2) and (3), respectively, and a network was considered "perfect" if for all input/output cases the error was smaller than 0.1 in the 0,1 case, or smaller than 0.2 in the ±1 case, respectively.

In Figure 3, the weights have been changed after every learning example, and all curves refer to an optimal choice of the only remaining parameter, $\varepsilon$ or $\eta$, respectively. For discrete as well as for continuous units, the ±1 networks again perform much better than their 0,1 counterparts. In the continuous case, the average learning times differ by about a factor of 7, and in the discrete case the discrepancy is even more pronounced. In addition, we observe that in ±1 networks learning with the generalized delta rule for discrete units is about twice as fast as with the backpropagation algorithm for continuous units.

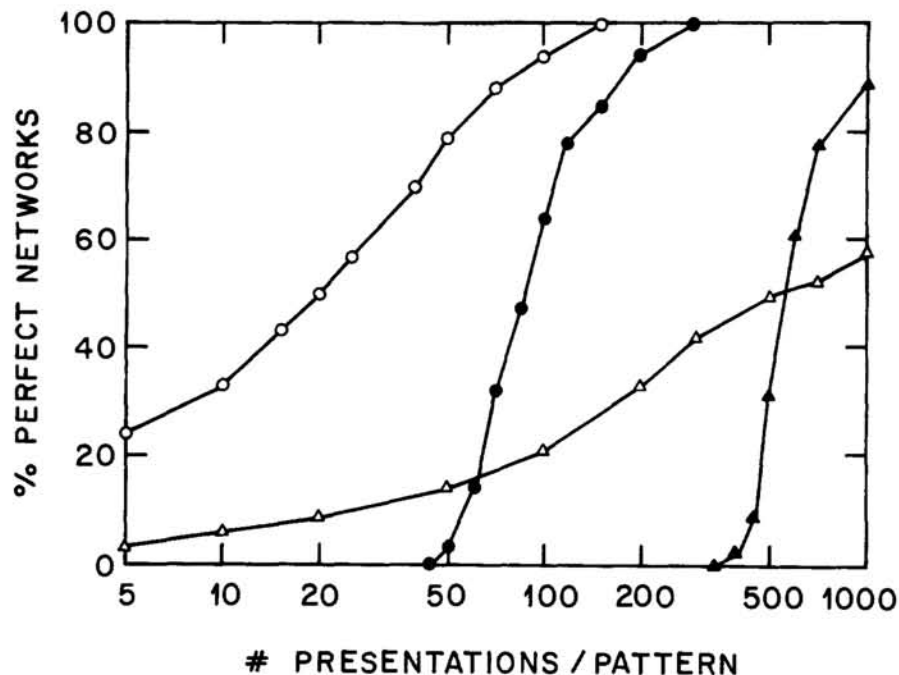

**Fig. 3.** Learning curves for an XOR-network with three hidden units (backpropagation/generalized delta rule, initial weights random, weights changed after every learning example). Open circles: discrete ±1 units, $\varepsilon = 0.05$; open triangles: discrete 0,1 units, $\varepsilon = 0.025$; full circles: continuous ±1 units, $\eta = 0.125$; full triangles; continuous 0,1 units, $\eta = 0.25$.

In Figure 4, the weight changes are accumulated over all 4 input/output cases, and only networks with continuous units are considered. Also in this case, the ±1 units lead to an improved learning behavior (the optimal $\eta$-values are about 2.5 and 5.0, respectively). They not only lead to significantly smaller learning times, but ±1 networks also appear to be less sensitive with respect to a variation of $\eta$ than the corresponding 0,1 versions.

The better performance of the ±1 models with continuous units can partly be attributed to the steeper slope of the chosen activation function, Eq. (3). A comparison with activation functions that have the same slope, however, shows that the networks with ±1 units still perform significantly better than those with 0,1 units. If the weights are updated after every learning example, e.g., the reduction in learning time remains as large as a factor of 5. In the case of backpropagation learning, the main reason for the better performance of ±1 units thus seems to be related to the fact that the algorithm does not modify weights which emerge from a unit with value zero. Similar observations have been made by Stornetta and Huberman,[5] who further find that the discrepancies become even more pronounced if the network size is increased.

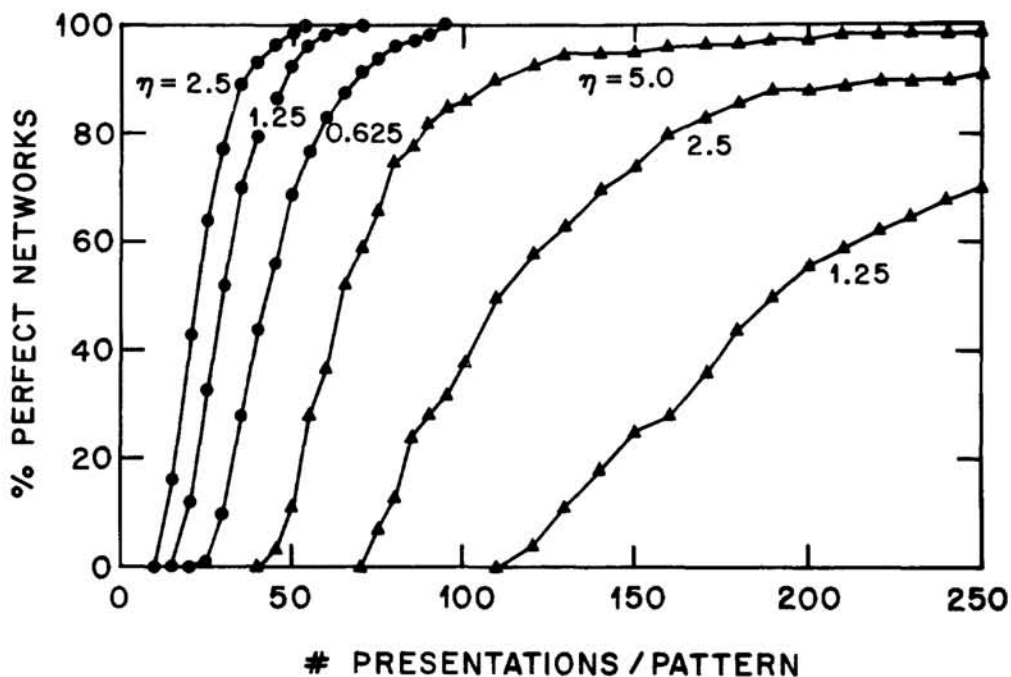

Fig. 4. Learning curves for an XOR-network with three hidden units (backpropagation, initial weights random, weight changes accumulated over all 4 input/output cases). Circles: continuous ±1 units; triangles: continuous 0,1 units.

In Figure 5, finally, we present results for a network that learns to detect mirror symmetry in the input pattern. The network consists of one output, one hidden, and four input units which are also directly connected to the output unit. We use the deterministic version of Boltzmann learning and change the weights after every presentation of a learning pattern. If the weights are allowed to assume arbitrary values, learning is rather slow and on average requires almost 700 presentations per pattern. We have observed, however, that the algorithm preferably seems to converge to solutions in which geometrically symmetric weights are opposite in sign and almost equal in magnitude (see also Ref. 3). This means that the symmetric input patterns are automatically treated as equivalent, as their net input to the hidden as well as to the output unit is zero. We have therefore investigated what happens if the weights are forced to be antisymmetric from the beginning. (The learning procedure, of course, has to be adjusted such that it preserves this antisymmetry). Figure 5 shows that such a problem-adapted weight-structure leads to a dramatic decrease in learning time.

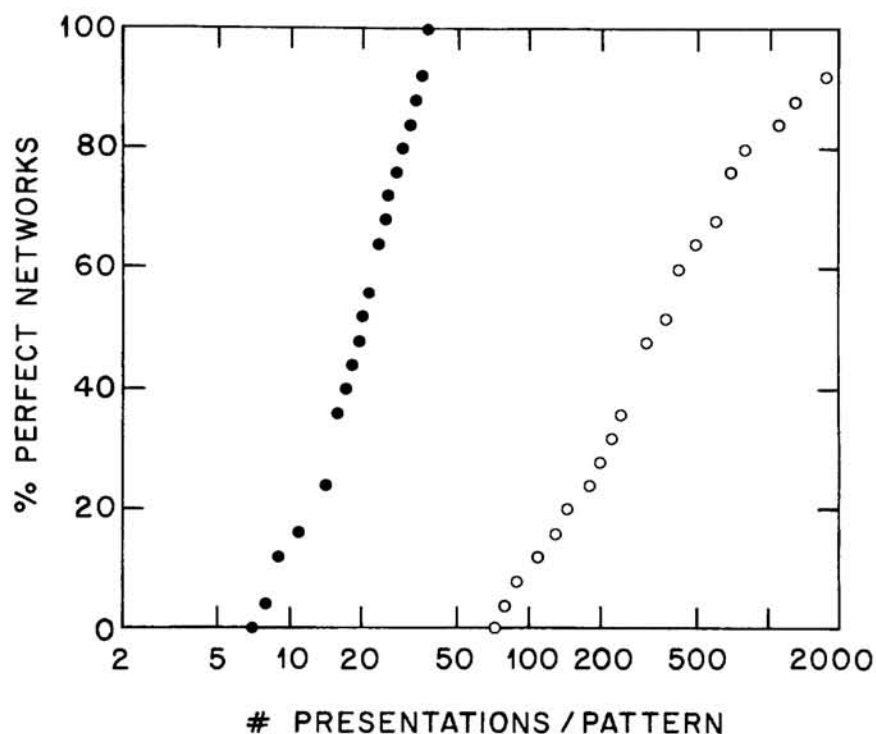

Fig. 5. Learning curves for a symmetry detection network with 4 input units and one hidden unit (deterministic Boltzmann learning, $\eta = 1$, discrete $\pm 1$ units, initial weights random, weights changed after every learning example). Full circles: symmetry-adapted weights; open circles: arbitrary weights, weight randomization ($p = 0.015$).

## CONCLUSIONS

The main results of our empirical study can be summarized as follows:
- Networks with ±1 units quite generally exhibit a significantly faster learning than the corresponding 0,1 versions.
- In addition, ±1 networks are often less sensitive to parameter variations than 0,1 networks.
- An adaptation of the weight-structure to the symmetries of the problem can lead to a drastic improvement of the learning behavior.

Our qualitative interpretations seem to indicate that the observed effects should not be restricted to the small examples considered in this paper. It would be very valuable, however, to have corresponding analytical results.

## REFERENCES

1. "Parallel Distributed Processing: Explorations in the Microstructure of Cognition", vol. 1: "Foundations", ed. by D.E. Rumelhart and J.L. McClelland (MIT Press, Cambridge), 1986, Chapters 7 & 8.
2. Y. le Cun, in "Disordered Systems and Biological Organization", ed. by E. Bienenstock, F. Fogelman Soulié, and G. Weisbuch (Springer, Berlin), 1986, pp. 233-240.
3. D.E. Rumelhart, G.E. Hinton, and R.J. Williams, Nature $\underline{323}$, 533 (1986).
4. M.L. Minsky and S. Papert, "Perceptrons" (MIT Press, Cambridge), 1969.
5. W.S. Stornetta and B.A. Huberman, IEEE Conference on "Neural Networks", San Diego, California, 21-24 June 1987.
